# A Convergent $O(n)$ Algorithm
# for Off-policy Temporal-difference Learning
# with Linear Function Approximation

**Richard S. Sutton, Csaba Szepesvári,**\* **Hamid Reza Maei**
Reinforcement Learning and Artificial Intelligence Laboratory
Department of Computing Science
University of Alberta
Edmonton, Alberta, Canada T6G 2E8

## Abstract

We introduce the first temporal-difference learning algorithm that is stable with linear function approximation and off-policy training, for any finite Markov decision process, behavior policy, and target policy, and whose complexity scales linearly in the number of parameters. We consider an i.i.d. policy-evaluation setting in which the data need not come from on-policy experience. The *gradient temporal-difference* (GTD) algorithm estimates the expected update vector of the TD(0) algorithm and performs stochastic gradient descent on its $L_2$ norm. We prove that this algorithm is stable and convergent under the usual stochastic approximation conditions to the same least-squares solution as found by the LSTD, but without LSTD's quadratic computational complexity. GTD is online and incremental, and does not involve multiplying by products of likelihood ratios as in importance-sampling methods.

## 1  Off-policy learning methods

Off-policy methods have an important role to play in the larger ambitions of modern reinforcement learning. In general, updates to a statistic of a dynamical process are said to be "off-policy" if their distribution does not match the dynamics of the process, particularly if the mismatch is due to the way actions are chosen. The prototypical example in reinforcement learning is the learning of the value function for one policy, the *target* policy, using data obtained while following another policy, the *behavior* policy. For example, the popular Q-learning algorithm (Watkins 1989) is an off-policy temporal-difference algorithm in which the target policy is greedy with respect to estimated action values, and the behavior policy is something more exploratory, such as a corresponding $\epsilon$-greedy policy. Off-policy methods are also critical to reinforcement-learning-based efforts to model human-level world knowledge and state representations as predictions of option outcomes (e.g., Sutton, Precup & Singh 1999; Sutton, Rafols & Koop 2006).

Unfortunately, off-policy methods such as Q-learning are not sound when used with approximations that are linear in the learned parameters—the most popular form of function approximation in reinforcement learning. Counterexamples have been known for many years (e.g., Baird 1995) in which Q-learning's parameters diverge to infinity for any positive step size. This is a severe problem in so far as function approximation is widely viewed as necessary for large-scale applications of reinforcement learning. The need is so great that practitioners have often simply ignored the problem and continued to use Q-learning with linear function approximation anyway. Although no instances

of absolute divergence in applications have been reported in the literature, the potential for instability is disturbing and probably belies real but less obvious problems.

The stability problem is not specific to reinforcement learning. Classical dynamic programming methods such as value and policy iteration are also off-policy methods and also diverge on some problems when used with linear function approximation. Reinforcement learning methods are actually an improvement over conventional dynamic programming methods in that at least they can be used stably with linear function approximation in their on-policy form. The stability problem is also not due to the interaction of control and prediction, or to stochastic approximation effects; the simplest counterexamples are for deterministic, expected-value-style, synchronous policy evaluation (see Baird 1995; Sutton & Barto 1998).

Prior to the current work, the possibility of instability could not be avoided whenever four individually desirable algorithmic features were combined: 1) off-policy updates, 2) temporal-difference learning, 3) linear function approximation, and 4) linear complexity in memory and per-time-step computation. If any one of these four is abandoned, then stable methods can be obtained relatively easily. But each feature brings value and practitioners are loath to give any of them up, as we discuss later in a penultimate related-work section. In this paper we present the first algorithm to achieve all four desirable features and be stable and convergent for all finite Markov decision processes, all target and behavior policies, and all feature representations for the linear approximator. Moreover, our algorithm does not use importance sampling and can be expected to be much better conditioned and of lower variance than importance sampling methods. Our algorithm can be viewed as performing stochastic gradient-descent in a novel objective function whose optimum is the least-squares TD solution. Our algorithm is also incremental and suitable for online use just as are simple temporal-difference learning algorithms such as Q-learning and TD($\lambda$) (Sutton 1988). Our algorithm can be broadly characterized as a gradient-descent version of TD(0), and accordingly we call it GTD(0).

## 2 Sub-sampling and i.i.d. formulations of temporal-difference learning

In this section we formulate the off-policy policy-evaluation problem for one-step temporal-difference learning such that the data consists of independent, identically-distributed (i.i.d.) samples. We start by considering the standard reinforcement learning framework, in which a learning agent interacts with an environment consisting of a finite Markov decision process (MDP). At each of a sequence of discrete time steps, $t = 1, 2, \ldots$, the environment is in a state $s_t \in \mathcal{S}$, the agent chooses an action $a_t \in \mathcal{A}$, and then the environment emits a reward $r_t \in \mathbb{R}$, and transitions to its next state $s_{t+1} \in \mathcal{S}$. The state and action sets are finite. State transitions are stochastic and dependent on the immediately preceding state and action. Rewards are stochastic and dependent on the preceding state and action, and on the next state. The agent process generating the actions is termed the *behavior policy*. To start, we assume a deterministic target policy $\pi : \mathcal{S} \rightarrow \mathcal{A}$. The objective is to learn an approximation to its state-value function:

$$V^\pi(s) = \mathbb{E}_\pi \left[ \sum_{t=1}^\infty \gamma^{t-1} r_t | s_1 = s \right], \tag{1}$$

where $\gamma \in [0, 1)$ is the *discount rate*. The learning is to be done without knowledge of the process dynamics and from observations of a single continuous trajectory with no resets.

In many problems of interest the state set is too large for it to be practical to approximate the value of each state individually. Here we consider linear function approximation, in which states are mapped to feature vectors with fewer components than the number of states. That is, for each state $s \in \mathcal{S}$ there is a corresponding feature vector $\phi(s) \in \mathbb{R}^n$, with $n \ll |\mathcal{S}|$. The approximation to the value function is then required to be linear in the feature vectors and a corresponding parameter vector $\theta \in \mathbb{R}^n$:

$$V^\pi(s) \approx \theta^\top \phi(s). \tag{2}$$

Further, we assume that the states $s_t$ are not visible to the learning agent in any way other than through the feature vectors. Thus this function approximation formulation can include partial-observability formulations such as POMDPs as a special case.

The environment and the behavior policy together generate a stream of states, actions and rewards, $s_1, a_1, r_1, s_2, a_2, r_2, \ldots$, which we can break into causally related 4-tuples, $(s_1, a_1, r_1, s_1')$,

$(s_2, a_2, r_2, s_2'), \ldots$, where $s_t' = s_{t+1}$. For some tuples, the action will match what the target policy would do in that state, and for others it will not. We can discard all of the latter as not relevant to the target policy. For the former, we can discard the action because it can be determined from the state via the target policy. With a slight abuse of notation, let $s_k$ denote the $k$th state in which an on-policy action was taken, and let $r_k$ and $s_k'$ denote the associated reward and next state. The $k$th on-policy transition, denoted $(s_k, r_k, s_k')$, is a triple consisting of the starting state of the transition, the reward on the transition, and the ending state of the transition. The corresponding data available to the learning algorithm is the triple $(\phi(s_k), r_k, \phi(s_k'))$.

The MDP under the behavior policy is assumed to be ergodic, so that it determines a stationary state-occupancy distribution $\mu(s) = \lim_{k \to \infty} Pr\{s_k = s\}$. For any state $s$, the MDP and target policy together determine an $N \times N$ state-transition-probability matrix $P$, where $p_{ss'} = Pr\{s_k' = s' | s_k = s\}$, and an $N \times 1$ expected-reward vector $R$, where $R_s = \mathbb{E}[r_k | s_k = s]$. These two together completely characterize the statistics of on-policy transitions, and all the samples in the sequence of $(\phi(s_k), r_k, \phi(s_k'))$ respect these statistics. The problem still has a Markov structure in that there are temporal dependencies between the sample transitions. In our analysis we first consider a formulation without such dependencies, the i.i.d. case, and then prove that our results extend to the original case.

In the i.i.d. formulation, the states $s_k$ are generated independently and identically distributed according to an arbitrary probability distribution $\mu$. From each $s_k$, a corresponding $s_k'$ is generated according to the on-policy state-transition matrix, $P$, and a corresponding $r_k$ is generated according to an arbitrary bounded distribution with expected value $R_{s_k}$. The final i.i.d. data sequence, from which an approximate value function is to be learned, is then the sequence $(\phi(s_k), r_k, \phi(s_k'))$, for $k = 1, 2, \ldots$ Further, because each sample is i.i.d., we can remove the indices and talk about a single tuple of random variables $(\phi, r, \phi')$ drawn from $\mu$.

It remains to define the objective of learning. The TD error for the linear setting is

$$\delta = r + \gamma \theta^\top \phi' - \theta^\top \phi. \tag{3}$$

Given this, we define the *one-step linear TD solution* as any value of $\theta$ at which

$$0 = \mathbb{E}[\delta\phi] = -A\theta + b, \tag{4}$$

where $A = \mathbb{E}\big[\phi(\phi - \gamma\phi')^\top\big]$ and $b = \mathbb{E}[r\phi]$. This is the parameter value to which the linear TD(0) algorithm (Sutton 1988) converges under on-policy training, as well as the value found by LSTD(0) (Bradtke & Barto 1996) under both on-policy and off-policy training. The TD solution is always a fixed-point of the linear TD(0) algorithm, but under off-policy training it may not be stable; if $\theta$ does not exactly satisfy (4), then the TD(0) algorithm may cause it to move away in expected value and eventually diverge to infinity.

## 3   The GTD(0) algorithm

We next present the idea and gradient-descent derivation leading to the GTD(0) algorithm. As discussed above, the vector $\mathbb{E}[\delta\phi]$ can be viewed as an error in the current solution $\theta$. The vector should be zero, so its norm is a measure of how far we are away from the TD solution. A distinctive feature of our gradient-descent analysis of temporal-difference learning is that we use as our objective function the $L_2$ norm of this vector:

$$J(\theta) = \mathbb{E}[\delta\phi]^\top \mathbb{E}[\delta\phi]. \tag{5}$$

This objective function is quadratic and unimodal; it's minimum value of 0 is achieved when $\mathbb{E}[\delta\phi] = 0$, which can always be achieved. The gradient of this objective function is

$$
\begin{aligned}
\nabla_\theta J(\theta) &= 2(\nabla_\theta \mathbb{E}[\delta\phi])\mathbb{E}[\delta\phi] \\
&= 2\mathbb{E}\big[\phi(\nabla_\theta\delta)^\top\big]^\top \mathbb{E}[\delta\phi] \\
&= -2\mathbb{E}\big[\phi(\phi - \gamma\phi')^\top\big]^\top \mathbb{E}[\delta\phi]. 
\end{aligned} \tag{6}
$$

This last equation is key to our analysis. We would like to take a stochastic gradient-descent approach, in which a small change is made on each sample in such a way that the expected update

is the direction opposite to the gradient. This is straightforward if the gradient can be written as a single expected value, but here we have a product of two expected values. One cannot sample both of them because the sample product will be biased by their correlation. However, one could store a long-term, quasi-stationary estimate of either of the expectations and then sample the other. The question is, which expectation should be estimated and stored, and which should be sampled? Both ways seem to lead to interesting learning algorithms.

First let us consider the algorithm obtained by forming and storing a separate estimate of the first expectation, that is, of the matrix $A = \mathbb{E}\big[\phi(\phi - \gamma\phi')^\top\big]$. This matrix is straightforward to estimate from experience as a simple arithmetic average of all previously observed sample outer products $\phi(\phi - \gamma\phi')^\top$. Note that $A$ is a stationary statistic in any fixed-policy policy-evaluation problem; it does not depend on $\theta$ and would not need to be re-estimated if $\theta$ were to change. Let $A_k$ be the estimate of $A$ after observing the first $k$ samples, $(\phi_1, r_1, \phi'_1), \ldots, (\phi_k, r_k, \phi'_k)$. Then this algorithm is defined by

$$A_k = \frac{1}{k} \sum_{i=1}^{k} \phi_i (\phi_i - \gamma\phi'_i)^\top \tag{7}$$

along with the gradient descent rule:

$$\theta_{k+1} = \theta_k + \alpha_k A_k^\top \delta_k \phi_k, \quad k \geq 1, \tag{8}$$

where $\theta_1$ is arbitrary, $\delta_k = r_k + \gamma\theta_k^\top\phi'_k - \theta_k^\top\phi_k$, and $\alpha_k > 0$ is a series of step-size parameters, possibly decreasing over time. We call this algorithm $A^\top$TD(0) because it is essentially conventional TD(0) prefixed by an estimate of the matrix $A^\top$. Although we find this algorithm interesting, we do not consider it further here because it requires $O(n^2)$ memory and computation per time step.

The second path to a stochastic-approximation algorithm for estimating the gradient (6) is to form and store an estimate of the second expectation, the vector $\mathbb{E}[\delta\phi]$, and to sample the first expectation, $\mathbb{E}\big[\phi(\phi - \gamma\phi')^\top\big]$. Let $u_k$ denote the estimate of $\mathbb{E}[\delta\phi]$ after observing the first $k - 1$ samples, with $u_1 = 0$. The GTD(0) algorithm is defined by

$$u_{k+1} = u_k + \beta_k (\delta_k \phi_k - u_k) \tag{9}$$

and

$$\theta_{k+1} = \theta_k + \alpha_k (\phi_k - \gamma\phi'_k)\phi_k^\top u_k, \tag{10}$$

where $\theta_1$ is arbitrary, $\delta_k$ is as in (3) using $\theta_k$, and $\alpha_k > 0$ and $\beta_k > 0$ are step-size parameters, possibly decreasing over time. Notice that if the product is formed right-to-left, then the entire computation is $O(n)$ per time step.

## 4 Convergence

The purpose of this section is to establish that GTD(0) converges with probability one to the TD solution in the i.i.d. problem formulation under standard assumptions. In particular, we have the following result:

**Theorem 4.1** (Convergence of GTD(0)). *Consider the GTD(0) iteration (9,10) with step-size sequences $\alpha_k$ and $\beta_k$ satisfying $\beta_k = \eta\alpha_k$, $\eta > 0$, $\alpha_k, \beta_k \in (0, 1]$, $\sum_{k=0}^{\infty} \alpha_k = \infty$, $\sum_{k=0}^{\infty} \alpha_k^2 < \infty$. Further assume that $(\phi_k, r_k, \phi'_k)$ is an i.i.d. sequence with uniformly bounded second moments. Let $A = \mathbb{E}\big[\phi_k(\phi_k - \gamma\phi'_k)^\top\big]$ and $b = \mathbb{E}[r_k\phi_k]$ (note that $A$ and $b$ are well-defined because the distribution of $(\phi_k, r_k, \phi'_k)$ does not depend on the sequence index $k$). Assume that $A$ is non-singular. Then the parameter vector $\theta_k$ converges with probability one to the TD solution (4).*

*Proof.* We use the ordinary-differential-equation (ODE) approach (Borkar & Meyn 2000). First, we rewrite the algorithm's two iterations as a single iteration in a combined parameter vector with $2n$ components $\rho_k^\top = (v_k^\top, \theta_k^\top)$, where $v_k = u_k/\sqrt{\eta}$, and a new reward-related vector with $2n$ components $g_{k+1}^\top = (r_k\phi_k^\top, 0^\top)$:

$$\rho_{k+1} = \rho_k + \alpha_k\sqrt{\eta}\,(G_{k+1}\rho_k + g_{k+1}),$$

where

$$G_{k+1} = \begin{pmatrix} -\sqrt{\eta}I & \phi_k(\gamma\phi'_k - \phi_k)^\top \\ (\phi_k - \gamma\phi'_k)\phi_k^\top & 0 \end{pmatrix}.$$

Let $G = \mathbb{E}[G_k]$ and $g = \mathbb{E}[g_k]$. Note that $G$ and $g$ are well-defined as by the assumption the process $\{\phi_k, r_k, \phi'_k\}_k$ is i.i.d. In particular,

$$G = \begin{pmatrix} -\sqrt{\eta}\, I & -A \\ A^\top & 0 \end{pmatrix}, \quad g = \begin{pmatrix} b \\ 0 \end{pmatrix}.$$

Further, note that (4) follows from

$$G\rho + g = 0, \tag{11}$$

where $\rho^\top = (v^\top, \theta^\top)$.

Now we apply Theorem 2.2 of Borkar & Meyn (2000). For this purpose we write $\rho_{k+1} = \rho_k + \alpha_k\sqrt{\eta}(G\rho_k + g + (G_{k+1} - G)\rho_k + (g_{k+1} - g)) = \rho_k + \alpha'_k(h(\rho_k) + M_{k+1})$, where $\alpha'_k = \alpha_k\sqrt{\eta}$, $h(\rho) = g + G\rho$ and $M_{k+1} = (G_{k+1} - G)\rho_k + g_{k+1} - g$. Let $\mathcal{F}_k = \sigma(\rho_1, M_1, \ldots, \rho_{k-1}, M_k)$. Theorem 2.2 requires the verification of the following conditions: (i) The function $h$ is Lipschitz and $h_\infty(\rho) = \lim_{r\to\infty} h(r\rho)/r$ is well-defined for every $\rho \in \mathbb{R}^{2n}$; (ii-a) The sequence $(M_k, \mathcal{F}_k)$ is a martingale difference sequence, and (ii-b) for some $C_0 > 0$, $\mathbb{E}[\|M_{k+1}\|^2 \,|\, \mathcal{F}_k] \leq C_0(1 + \|\rho_k\|^2)$ holds for any initial parameter vector $\rho_1$; (iii) The sequence $\alpha'_k$ satisfies $0 < \alpha'_k \leq 1$, $\sum_{k=1}^\infty \alpha'_k = \infty$, $\sum_{k=1}^\infty (\alpha'_k)^2 < +\infty$; and (iv) The ODE $\dot{\rho} = h(\rho)$ has a globally asymptotically stable equilibrium. Clearly, $h(\rho)$ is Lipschitz with coefficient $\|G\|$ and $h_\infty(\rho) = G\rho$. By construction, $(M_k, \mathcal{F}_k)$ satisfies $\mathbb{E}[M_{k+1}|\mathcal{F}_k] = 0$ and $M_k \in \mathcal{F}_k$, i.e., it is a martingale difference sequence. Condition (ii-b) can be shown to hold by a simple application of the triangle inequality and the boundedness of the the the second moments of $(\phi_k, r_k, \phi'_k)$. Condition (iii) is satisfied by our conditions on the step-size sequences $\alpha_k, \beta_k$. Finally, the last condition (iv) will follow from the elementary theory of linear differential equations if we can show that the real parts of all the eigenvalues of $G$ are negative.

First, let us show that $G$ is non-singular. Using the determinant rule for partitioned matrices[1] we get $\det(G) = \det(A^\top A) \neq 0$. This indicates that all the eigenvalues of $G$ are non-zero. Now, let $\lambda \in \mathbb{C}$, $\lambda \neq 0$ be an eigenvalue of $G$ with corresponding normalized eigenvector $x \in \mathbb{C}^{2n}$; that is, $\|x\|^2 = x^* x = 1$, where $x^*$ is the complex conjugate of $x$. Hence $x^* G x = \lambda$. Let $x^\top = (x_1^\top, x_2^\top)$, where $x_1, x_2 \in \mathbb{C}^n$. Using the definition of $G$, $\lambda = x^* G x = -\sqrt{\eta}\|x_1\|^2 + x_1^* A x_2 - x_2^* A^\top x_1$. Because $A$ is real, $A^* = A^\top$, and it follows that $(x_1^* A x_2)^* = x_2^* A^\top x_1$. Thus, $\mathrm{Re}(\lambda) = \mathrm{Re}(x^* G x) = -\sqrt{\eta}\|x_1\|^2 \leq 0$. We are now done if we show that $x_1$ cannot be zero. If $x_1 = 0$, then from $\lambda = x^* G x$ we get that $\lambda = 0$, which contradicts with $\lambda \neq 0$. $\qquad\square$

The next result concerns the convergence of GTD(0) when $(\phi_k, r_k, \phi'_k)$ is obtained by the off-policy sub-sampling process described originally in Section 2. We make the following assumption:

**Assumption A1** The behavior policy $\pi_b$ (generator of the actions $a_t$) selects all actions of the target policy $\pi$ with positive probability in every state, and the target policy is deterministic.

This assumption is needed to ensure that the sub-sampled process $s_k$ is well-defined and that the obtained sample is of "high quality". Under this assumption it holds that $s_k$ is again a Markov chain by the strong Markov property of Markov processes (as the times selected when actions correspond to those of the behavior policy form Markov times with respect to the filtration defined by the original process $s_t$). The following theorem shows that the conclusion of the previous result continues to hold in this case:

**Theorem 4.2** (Convergence of GTD(0) with a sub-sampled process.)**.** *Assume A1. Let the parameters $\theta_k, u_k$ be updated by (9,10). Further assume that $(\phi_k, r_k, \phi'_k)$ is such that $\mathbb{E}[\|\phi_k\|^2|s_{k-1}]$, $\mathbb{E}[r_k^2|s_{k-1}]$, $\mathbb{E}[\|\phi'_k\|^2|s_{k-1}]$ are uniformly bounded. Assume that the Markov chain $(s_k)$ is aperiodic and irreducible, so that $\lim_{k\to\infty} \mathbb{P}(s_k = s'|s_0 = s) = \mu(s')$ exists and is unique. Let $s$ be a state randomly drawn from $\mu$, and let $s'$ be a state obtained by following $\pi$ for one time step in the MDP from $s$. Further, let $r(s, s')$ be the reward incurred. Let $A = \mathbb{E}[\phi(s)(\phi(s) - \gamma\phi(s'))^\top]$ and $b = \mathbb{E}[r(s, s')\phi(s)]$. Assume that $A$ is non-singular. Then the parameter vector $\theta_k$ converges with probability one to the TD solution (4), provided that $s_1 \sim \mu$.*

*Proof.* The proof of Theorem 4.1 goes through without any changes once we observe that $G = \mathbb{E}[G_{k+1}|\mathcal{F}_k]$ and $g = \mathbb{E}[g_{k+1}\,|\,\mathcal{F}_k]$. $\qquad\square$

The condition that $(s_k)$ is aperiodic and irreducible guarantees the existence of the steady state distribution $\mu$. Further, the aperiodicity and irreducibility of $(s_k)$ follows from the same property of the original process $(s_t)$. For further discussion of these conditions cf. Section 6.3 of Bertsekas and Tsitsiklis (1996).

With considerable more work the result can be extended to the case when $s_1$ follows an arbitrary distribution. This requires an extension of Theorem 2.2 of Borkar and Meyn (2000) to processes of the form $\rho_{k+1} + \rho_k\big(h(\rho_k) + M_{k+1} + e_{k+1}\big)$, where $e_{k+1}$ is a fast decaying perturbation (see, e.g., the proof of Proposition 4.8 of Bertsekas and Tsitsiklis (1996)).

## 5 Extensions to action values, stochastic target policies, and other sample weightings

The GTD algorithm extends immediately to the case of off-policy learning of action-value functions. For this assume that a behavior policy $\pi_b$ is followed that samples all actions in every state with positive probability. Let the target policy to be evaluated be $\pi$. In this case the basis functions are dependent on both the states and actions: $\phi : S \times A \to \mathbb{R}^n$. The learning equations are unchanged, except that $\phi_t$ and $\phi'_t$ are redefined as follows:

$$\phi_t = \phi(s_t, a_t), \tag{12}$$

$$\phi'_t = \sum_a \pi(s_{t+1}, a)\phi(s_{t+1}, a). \tag{13}$$

(We use time indices $t$ denoting physical time.) Here $\pi(s, a)$ is the probability of selecting action $a$ in state $s$ under the target policy $\pi$. Let us call the resulting algorithm "one-step gradient-based Q-evaluation," or GQE(0).

**Theorem 5.1** (Convergence of GQE(0)). *Assume that $s_t$ is a state sequence generated by following some stationary policy $\pi_b$ in a finite MDP. Let $r_t$ be the corresponding sequence of rewards and let $\phi_t, \phi'_t$ be given by the respective equations* (12) *and* (13)*, and assume that $\mathbb{E}\big[\|\phi_t\|^2 | s_{t-1}\big]$, $\mathbb{E}\big[r_t^2 | s_{t-1}\big]$, $\mathbb{E}\big[\|\phi'_t\|^2 | s_{t-1}\big]$ are uniformly bounded. Let the parameters $\theta_t, u_t$ be updated by Equations* (9) *and* (10)*. Assume that the Markov chain $(s_t)$ is aperiodic and irreducible, so that $\lim_{t\to\infty} \mathbb{P}(s_t = s' | s_0 = s) = \mu(s')$ exists and is unique. Let $s$ be a state randomly drawn from $\mu$, $a$ be an action chosen by $\pi_b$ in $s$, let $s'$ be the next state obtained and let $a' = \pi(s')$ be the action chosen by the target policy in state $s'$. Further, let $r(s, a, s')$ be the reward incurred in this transition. Let $A = \mathbb{E}\big[\phi(s, a)(\phi(s, a) - \gamma\phi(s', a'))^\top\big]$ and $b = \mathbb{E}[r(s, a, s')\phi(s, a)]$. Assume that $A$ is non-singular. Then the parameter vector $\theta_t$ converges with probability one to a TD solution* (4)*, provided that $s_1$ is selected from the steady-state distribution $\mu$.*

The proof is almost identical to that of Theorem 4.2, and hence it is omitted.

Our main convergence results are also readily generalized to stochastic target policies by replacing the sub-sampling process described in Section 2 with a sample-weighting process. That is, instead of including or excluding transitions depending upon whether the action taken matches a deterministic policy, we include all transitions but give each a weight. For example, we might let the weight $w_t$ for time step $t$ be equal to the probability $\pi(s_t, a_t)$ of taking the action actually taken under the target policy. We can consider the i.i.d. samples now to have four components $(\phi_k, r_k, \phi'_k, w_k)$, with the update rules (9) and (10) replaced by

$$u_{k+1} = u_k + \beta_k(\delta_k\phi_k - u_k)w_k, \tag{14}$$

and

$$\theta_{k+1} = \theta_k + \alpha_k(\phi_k - \gamma\phi'_k)\phi_k^\top u_k w_k. \tag{15}$$

Each sample is also weighted by $w_k$ in the expected values, such as that defining the TD solution (4). With these changes, Theorems 4.1 and 4.2 go through immediately for stochastic policies. The reweighting is, in effect, an adjustment to the i.i.d. sampling distribution, $\mu$, and thus our results hold because they hold for all $\mu$. The choice $w_t = \pi(s_t, a_t)$ is only one possibility, notable for its equivalence to our original case if the target policy is deterministic. Another natural weighting is $w_t = \pi(s_t, a_t)/\pi_b(s_t, a_t)$, where $\pi_b$ is the behavior policy. This weighting may result in the TD solution (4) better matching the target policy's value function (1).

# 6 Related work

There have been several prior attempts to attain the four desirable algorithmic features mentioned at the beginning this paper (off-policy stability, temporal-difference learning, linear function approximation, and $O(n)$ complexity) but none has been completely successful.

One idea for retaining all four desirable features is to use importance sampling techniques to re-weight off-policy updates so that they are in the same direction as on-policy updates in expected value (Precup, Sutton & Dasgupta 2001; Precup, Sutton & Singh 2000). Convergence can sometimes then be assured by existing results on the convergence of on-policy methods (Tsitsiklis & Van Roy 1997; Tadic 2001). However, the importance sampling weights are cumulative products of (possibly many) target-to-behavior-policy likelihood ratios, and consequently they and the corresponding updates may be of very high variance. The use of "recognizers" to construct the target policy directly from the behavior policy (Precup, Sutton, Paduraru, Koop & Singh 2006) is one strategy for limiting the variance; another is careful choice of the target policies (see Precup, Sutton & Dasgupta 2001). However, it remains the case that for all of such methods to date there are always choices of problem, behavior policy, and target policy for which the variance is *infinite*, and thus for which there is no guarantee of convergence.

Residual gradient algorithms (Baird 1995) have also been proposed as a way of obtaining all four desirable features. These methods can be viewed as gradient descent in the expected squared TD error, $\mathbb{E}[\delta^2]$; thus they converge stably to the solution that minimizes this objective for arbitrary differentiable function approximators. However, this solution has always been found to be much inferior to the TD solution (exemplified by (4) for the one-step linear case). In the literature (Baird 1995; Sutton & Barto 1998), it is often claimed that residual-gradient methods are guaranteed to find the TD solution in two special cases: 1) systems with deterministic transitions and 2) systems in which two samples can be drawn for each next state (e.g., for which a simulation model is available). Our own analysis indicates that even these two special requirements are insufficient to guarantee convergence to the TD solution.[2]

Gordon (1995) and others have questioned the need for linear function approximation. He has proposed replacing linear function approximation with a more restricted class of approximators, known as *averagers*, that never extrapolate outside the range of the observed data and thus cannot diverge. Rightly or wrongly, averagers have been seen as being too constraining and have not been used on large applications involving online learning. Linear methods, on the other hand, have been widely used (e.g., Baxter, Tridgell & Weaver 1998; Sturtevant & White 2006; Schaeffer, Hlynka & Jussila 2001).

The need for linear complexity has also been questioned. Second-order methods for linear approximators, such as LSTD (Bradtke & Barto 1996; Boyan 2002) and LSPI (Lagoudakis & Parr 2003; see also Peters, Vijayakumar & Schaal 2005), can be effective on moderately sized problems. If the number of features in the linear approximator is $n$, then these methods require memory and per-time-step computation that is $O(n^2)$. Newer incremental methods such as iLSTD (Geramifard, Bowling & Sutton 2006) have reduced the per-time-complexity to $O(n)$, but are still $O(n^2)$ in memory. Sparsification methods may reduce the complexity further, they do not help in the general case, and may apply to $O(n)$ methods as well to further reduce their complexity. Linear function approximation is most powerful when very large numbers of features are used, perhaps *millions* of features (e.g., as in Silver, Sutton & Müller 2007). In such cases, $O(n^2)$ methods are not feasible.

# 7 Conclusion

GTD(0) is the first off-policy TD algorithm to converge under general conditions with linear function approximation and linear complexity. As such, it breaks new ground in terms of important,

absolute abilities not previous available in existing algorithms. We have conducted empirical studies with the GTD(0) algorithm and have confirmed that it converges reliably on standard off-policy counterexamples such as Baird's (1995) "star" problem. On on-policy problems such as the $n$-state random walk (Sutton 1988; Sutton & Barto 1998), GTD(0) does not seem to learn as efficiently as classic TD(0), although we are still exploring different ways of setting the step-size parameters, and other variations on the algorithm. It is not clear that the GTD(0) algorithm in its current form will be a fully satisfactory solution to the off-policy learning problem, but it is clear that is breaks new ground and achieves important abilities that were previously unattainable.

### Acknowledgments

The authors gratefully acknowledge insights and assistance they have received from David Silver, Eric Wiewiora, Mark Ring, Michael Bowling, and Alborz Geramifard. This research was supported by iCORE, NSERC and the Alberta Ingenuity Fund.

## Footnotes

\*Csaba Szepesvári is on leave from MTA SZTAKI.

[1]According to this rule, if $A \in \mathbb{R}^{n\times n}$, $B \in \mathbb{R}^{n\times m}$, $C \in \mathbb{R}^{m\times n}$, $D \in \mathbb{R}^{m\times m}$ then for $F = [A\ B;\ C\ D] \in \mathbb{R}^{(n+m)\times(n+m)}$, $\det(F) = \det(A)\det(D - CA^{-1}B)$.

[2]For a counterexample, consider that given in Dayan's (1992) Figure 2, except now consider that state A is actually two states, A and A', which share the same feature vector. The two states occur with 50-50 probability, and when one occurs the transition is always deterministically to B followed by the outcome 1, whereas when the other occurs the transition is always deterministically to the outcome 0. In this case $V(A)$ and $V(B)$ will converge under the residual-gradient algorithm to the wrong answers, $1/3$ and $2/3$, even though the system is deterministic, and even if multiple samples are drawn from each state (they will all be the same).

### References

Baird, L. C. (1995). Residual algorithms: Reinforcement learning with function approximation. In *Proceedings of the Twelfth International Conference on Machine Learning*, pp. 30–37. Morgan Kaufmann.

Baxter, J., Tridgell, A., Weaver, L. (1998). Experiments in parameter learning using temporal differences. *International Computer Chess Association Journal*, *21*, 84–99.

Bertsekas, D. P., Tsitsiklis. J. (1996). *Neuro-Dynamic Programming*. Athena Scientific, 1996.

Borkar, V. S. and Meyn, S. P. (2000). The ODE method for convergence of stochastic approximation and reinforcement learning. *SIAM Journal on Control And Optimization* , 38(2):447–469.

Boyan, J. (2002). Technical update: Least-squares temporal difference learning. *Machine Learning*, 49:233–246.

Bradtke, S., Barto, A. G. (1996). Linear least-squares algorithms for temporal difference learning. *Machine Learning*, 22:33–57.

Dayan, P. (1992). The convergence of TD($\lambda$) for general $\lambda$. *Machine Learning*, 8:341–362.

Geramifard, A., Bowling, M., Sutton, R. S. (2006). Incremental least-square temporal difference learning. *Proceedings of the National Conference on Artificial Intelligence*, pp. 356–361.

Gordon, G. J. (1995). Stable function approximation in dynamic programming. *Proceedings of the Twelfth International Conference on Machine Learning*, pp. 261–268. Morgan Kaufmann, San Francisco.

Lagoudakis, M., Parr, R. (2003). Least squares policy iteration. *Journal of Machine Learning Research*, 4:1107-1149.

Peters, J., Vijayakumar, S. and Schaal, S. (2005). Natural Actor-Critic. *Proceedings of the 16th European Conference on Machine Learning*, pp. 280–291.

Precup, D., Sutton, R. S. and Dasgupta, S. (2001). Off-policy temporal-difference learning with function approximation. *Proceedings of the 18th International Conference on Machine Learning*, pp. 417–424.

Precup, D., Sutton, R. S., Paduraru, C., Koop, A., Singh, S. (2006). Off-policy Learning with Recognizers. *Advances in Neural Information Processing Systems 18*.

Precup, D., Sutton, R. S., Singh, S. (2000). Eligibility traces for off-policy policy evaluation. *Proceedings of the 17th International Conference on Machine Learning*, pp. 759–766. Morgan Kaufmann.

Schaeffer, J., Hlynka, M., Jussila, V. (2001). Temporal difference learning applied to a high-performance game-playing program. *Proceedings of the International Joint Conference on Artificial Intelligence*, pp. 529–534.

Silver, D., Sutton, R. S., Müller, M. (2007). Reinforcement learning of local shape in the game of Go. *Proceedings of the 20th International Joint Conference on Artificial Intelligence*, pp. 1053–1058.

Sturtevant, N. R., White, A. M. (2006). Feature construction for reinforcement learning in hearts. In *Proceedings of the 5th International Conference on Computers and Games*.

Sutton, R. S. (1988). Learning to predict by the method of temporal differences. *Machine Learning*, 3:9–44.

Sutton, R. S., Barto, A. G. (1998). *Reinforcement Learning: An Introduction*. MIT Press.

Sutton, R.S., Precup D. and Singh, S (1999). Between MDPs and semi-MDPs: A framework for temporal abstraction in reinforcement learning. *Artificial Intelligence*, 112:181–211.

Sutton, R. S., Rafols, E.J., and Koop, A. 2006. Temporal abstraction in temporal-difference networks. *Advances in Neural Information Processing Systems 18*.

Tadic, V. (2001). On the convergence of temporal-difference learning with linear function approximation. In *Machine Learning* 42:241–267

Tsitsiklis, J. N., and Van Roy, B. (1997). An analysis of temporal-difference learning with function approximation. *IEEE Transactions on Automatic Control*, 42:674–690.

Watkins, C. J. C. H. (1989). *Learning from Delayed Rewards*. Ph.D. thesis, Cambridge University.

